# Fast Computation of Posterior Mode in Multi-Level Hierarchical Models

**Liang Zhang**
Department of Statistical Science
Duke University
Durham, NC 27708
lz9@stat.duke.edu

**Deepak Agarwal**
Yahoo! Research
2821 Mission College Blvd.
Santa Clara, CA 95054
dagarwal@yahoo-inc.com

## Abstract

Multi-level hierarchical models provide an attractive framework for incorporating correlations induced in a response variable that is organized hierarchically. Model fitting is challenging, especially for a hierarchy with a large number of nodes. We provide a novel algorithm based on a multi-scale Kalman filter that is both scalable and easy to implement. For Gaussian response, we show our method provides the maximum a-posteriori (MAP) parameter estimates; for non-Gaussian response, parameter estimation is performed through a Laplace approximation. However, the Laplace approximation provides biased parameter estimates that is corrected through a parametric bootstrap procedure. We illustrate through simulation studies and analyses of real world data sets in health care and online advertising.

## 1  Introduction

In many real-world prediction problems, the response variable of interest is clustered hierarchically. For instance, in studying the immunization status of a set of children in a particular geographic location, the children are naturally clustered by families, which in turn are clustered into communities. The clustering often induce correlations in the response variable; models that exploit this provide significant improvement in predictive performance. Multi-level hierarchical models provide an attractive framework for modeling such correlations. Although routinely applied to moderate sized data (few thousand nodes) in several fields like epidemiology, social sciences, biology [3], model fitting is computationally expensive and is usually performed through a Cholesky decomposition of a $q$ (number of nodes in the hierarchy) dimensional matrix. Recently, such models have shown promise in a novel application of internet advertising [1] where the goal is to select top-$k$ advertisements to be shown on a webpage to maximize the click-through rates. To capture the semantic meaning of content in a parsimonious way, it is commonplace to classify webpages and ads into large pre-defined hierarchies. The hierarchy in such applications consist of several levels and the total number of nodes may run into millions. Moreover, the main goal is to exploit the hierarchy for obtaining better predictions; computing the full posterior predictive distribution is of secondary importance. Existing fitting algorithms are difficult to implement and do not scale well for such problems. In this paper, we provide a novel, *fast* and *easy* to implement algorithm to compute the posterior mode of parameters for such models on datasets organized hierarchically into millions of nodes with several levels. The key component of our algorithm is a multi-scale Kalman filter that expedites the computation of an expensive to compute conditional posterior.

The central idea in multi-level hierarchical (MLH hereafter) models is "shrinkage" across the nodes in the hierarchy. More specifically, these models assume a multi-level prior wherein parameters of children nodes are assumed to be drawn from a distribution centered around the parameter of the parent. This bottom-up, recursive assumption provides a posterior whose estimates at the finest resolution are smoothed using data on the lineage path of the node in the hierarchy. The fundamental

| Notation | Meaning |
| --- | --- |
| $T_j$ | Level $j$ of the hierarchy $T$ |
| $m_j$ | The number of nodes at level $j$ in $T$ |
| $q$ | The total number of nodes in $T$ |
| $pa(r)$ | The parent node of node $r$ in $T$ |
| $c_i(r)$ | The $i$th child node of node $r$ in $T$ |
| $n_r$ | The number of observations at leaf node $r$ |
| $y_{ir}$ | The $i$th observation (response) at *leaf* node $r$ |
| $\boldsymbol{Y}$ | $\{y_{ir}, i = 1, \cdots, n_r, r \in T\}$ |
| $\boldsymbol{x}_{ir}$ | The $i$th observation ($p$-dimensional covariates) at *leaf* node $r$ |
| $\boldsymbol{X}$ | $\{\boldsymbol{x}_{ir}, i = 1, \cdots, n_r, r \in T\}$ |
| $\boldsymbol{\beta}$ | The regression parameter vector associated with $\boldsymbol{X}$ |
| $\phi_r^j$ | The random effect parameter at node $r$ at level $j$ |
| $\boldsymbol{\phi}$ | $\{\phi_r^j, r \in T, j = 1, \cdots, L\}$ |
| $V$ | The residual variance of $y_{ir}$, if $y_{ir}$ has a Gaussian model |
| $\gamma_j$ | The variance of $\phi_r^j$ for all the nodes at level $j$ |
| $\boldsymbol{\gamma}$ | $\{\gamma_1, \cdots, \gamma_L\}$ |
| $\phi_{r\|r}^j$ | The mean of $\phi_r^j \| \{y_{ir'}, i = 1, \cdots, n_{r'}, \forall r' \prec r\}$ |
| $\sigma_{r\|r}^j$ | The variance of $\phi_r^j \| \{y_{ir'}, i = 1, \cdots, n_{r'}, \forall r' \prec r\}$ |
| $\hat{\phi}_r^j$ | The mean of $\phi_r^j \| \{y_{ir'}, i = 1, \cdots, n_{r'}, \forall r' \in T_L\}$ |
| $\sigma_r^j$ | The variance of $\phi_r^j \| \{y_{ir'}, i = 1, \cdots, n_{r'}, \forall r' \in T_L\}$ |

Table 1: A list of the key notations.

assumption is that the hierarchy, determined from domain knowledge, provides a natural clustering to account for latent processes generating the data which, when incorporated into the model, improve predictions. Although MLH models are intuitive, parameter estimation presents a formidable challenge, especially for large hierarchies. For Gaussian response, the main computational bottleneck is the Cholesky factorization of a dense covariance matrix whose order depends on the number of nodes, this is expensive for large problems. For non-Gaussian response (e.g binary data), the non-quadratic nature of the log-likelihood adds on an additional challenge of approximating an integral whose dimension depends on the number of nodes in the hierarchy. This is an active area of research in statistics with several solutions being proposed, such as [5] (see references therein as well). For Cholesky factorization, techniques based on sparse factorization of the covariance matrix have been recently proposed in [5]. For non-Gaussian models, solutions require marginalization over a high dimensional integral and is often accomplished through higher order Taylor series approximations[6]. However, these techniques involve linear algebra that is often non-intuitive and difficult to implement. A more natural computational scheme that exploits the structure of the model is based on Gibbs sampling; however, it is not scalable due to slow convergence.

Our **contributions** are as follows: We provide a novel fitting procedure based on multi-scale Kalman filter algorithm that directly exploits the hierarchical structure of the problem and computes the posterior mode of MLH parameters. The complexity of our method is almost linear in the number of nodes in the hierarchy. Other than scalability, our fitting procedure is more intuitive and easy to implement. We note that although multi-scale Kalman filters have been studied in the electrical engineering literature [2] and spatial statistics, their application to fitting MLH is novel. Moreover, fitting such models to non-Gaussian data present formidable challenges as we illustrate in the paper. We provide strategies to overcome those through a bootstrap correction and compare with the commonly used cross-validation approach. Our methods are illustrated on simulated data, benchmark data and data obtained from an internet advertising application.

## 2 MLH for Gaussian Responses

Assume we have a hierarchy $T$ consisting of $L$ levels (root is level 0), for which $m_j, j = 0, \cdots, L$, denotes the number of nodes at level $j$. Denote the set of nodes at level $j$ in the hierarchy $T$ as $T_j$. For node $r$ in $T$, denote the parent of $r$ as $pa(r)$, and the $i^{th}$ child of node $r$ as $c_i(r)$. If a node $r'$ is

a descendent of $r$, we say $r' \prec r$. Since the hierarchy has $L$ levels, $T_L$ denotes the set of leaf nodes in the hierarchy. Let $y_{ir}, i = 1, \cdots, n_r$ denote the $i^{th}$ observation at *leaf* node $r$, and $\boldsymbol{x}_{ir}$ denote the $p$-dimensional covariate vector associated with $y_{ir}$. For simplicity, we assume all observations are available at leaf nodes (a more general case where each node in the hierarchy can have observations is easily obtained from our algorithm). Consider the Gaussian MLH defined by

$$y_{ir}|\phi_r^L \sim N(\boldsymbol{x}_{ir}^{'}\boldsymbol{\beta} + \phi_r^L, V), \tag{1}$$

where $\boldsymbol{\beta}$ is a fixed effect parameter vector and $\phi_r^j$ is a random effect associated with node $r$ at level $j$ with joint distribution defined through a set of hierarchical conditional distributions $p(\phi_r^j|\phi_{pa(r)}^{j-1}), j = 0, \cdots, L$, where $\phi_0^0 = 0$. The form of $p(\phi_r^j|\phi_{pa(r)}^{j-1}), j = 1, \cdots, L$ is assumed to be

$$\phi_r^j|\phi_{pa(r)}^{j-1} \sim N(\phi_{pa(r)}^{j-1}, \gamma_j); j = 1, \cdots, L, \tag{2}$$

where $\boldsymbol{\gamma} = (\gamma_1, \cdots, \gamma_L)$ is a vector of level-specific variance components that control the amount of smoothing. To complete the model specification in a Bayesian framework, we put a vague prior on $V$ ($\pi(V) \propto 1/V$) and a mild quadratic prior on $\gamma_i$ ($\pi(\gamma_i|V) \propto V/(V + \gamma_i)^2$). For $\boldsymbol{\beta}$, we assume a non-informative prior, i.e., $\pi(\boldsymbol{\beta}) \propto 1$.

The specification of MLH given by Equation 2 is referred to as the *centered* parametrization and was shown to provide good performance in a fully Bayesian framework by [9]. An equivalent way of specifying MLH is obtained by associating independent random variables $b_r^j \sim N(0, \gamma_j)$ to the nodes and replacing $\phi_r^L$ in (1) by the sum of the $b_r^j$ parameters along the lineage path from root to leaf node in the hierarchy. We denote this compactly as $\boldsymbol{z}_r'\boldsymbol{b}$, where $\boldsymbol{b}$ is a vector of $b_r^j$ for all the nodes in the hierarchy, and $\boldsymbol{z}_r$ is a vector of 0/1's turned on for nodes in the path of node $r$. More compactly, let $\boldsymbol{y} = \{y_{ir}, i = 1, \cdots, n_r, r \in T\}$, and $\boldsymbol{X}$ as well as $\boldsymbol{Z}$ be the corresponding matrix of vectors $\boldsymbol{x}_{ir}$ and $\boldsymbol{z}_r$ for $i = 1, \cdots n_r$ and $r \in T$, then $\boldsymbol{y} \sim N(\boldsymbol{X}'\boldsymbol{\beta} + \boldsymbol{Z}\boldsymbol{b}, V\boldsymbol{I})$ with $\boldsymbol{b} \sim N(\boldsymbol{0}, \Omega(\boldsymbol{\gamma}))$. The problem is to compute the posterior mode of $(\boldsymbol{\beta}_{p \times 1}, \boldsymbol{b}_{q \times 1}, \boldsymbol{\gamma}_{L \times 1}, V)$ where $q = \sum_{j=1}^{L} m_j$. The main computational bottleneck is computing the Cholesky factor of a $q \times q$ matrix $(\boldsymbol{Z}'\boldsymbol{Z} + \Omega^{-1})$, this is expensive for large values of $q$. Existing state-of-the-art methods are based on sparse Cholesky factorization; we provide a more direct way. In fact, our method provides a MAP estimate of the parameters for the Gaussian case. For non-Gaussian case, we provide an approximation to the MAP through the Laplace method coupled with a bootstrap correction. We also note that our method apply if the random effects are vectors and enter into equation (2) as linear combination with some covariate vector. In this paper, we illustrate through a scalar.

## 2.1 Model Fitting

Throughout, we work with the parametrization specified by $\phi$. The main component of our fitting algorithm is computing the conditional posterior distribution of $\phi = \{\phi_r^j, r \in T, j = 1, \cdots, L\}$ given $(\boldsymbol{\beta}, V, \boldsymbol{\gamma})$. Since the parameters $V$ and $\boldsymbol{\gamma}$ are unknown, we estimate them through an EM algorithm. The multi-scale Kalman filter (described next) computes the conditional posterior of $\phi$ mentioned above and is used in the inner loop of the EM.

As in temporal state space models, the Kalman filter consists of two steps - a)*Filtering*: where one propagates information from leaves to the root and b) *Smoothing*: where information is propagated from root all the way down to the leaves.

*Filtering:*

Denote the current estimates of $\boldsymbol{\beta}$, $\boldsymbol{\gamma}$ and $V$ as $\hat{\boldsymbol{\beta}}$, $\hat{\boldsymbol{\gamma}}$, and $\hat{V}$ respectively. Then, $e_{ir} = y_{ir} - \boldsymbol{x}_{ir}^{'}\hat{\boldsymbol{\beta}}$ are the residuals and $Var(\phi_r^j) = \Sigma_j = \sum_{i=1}^{j} \hat{\gamma}_i, r \in T_j$ are the marginal variances of the random effects. If the conditional posterior distribution $\phi_r^L|\{y_{ir}, i = 1, \cdots, n_r\} \sim N(\phi_{r|r}^L, \sigma_{r|r}^L)$, the first step is to update $\phi_{r|r}^L$ and $\sigma_{r|r}^L$ for all leaf random effects $\phi_r^L$ using standard Bayesian update formula for Gaussian models

$$\phi_{r|r}^L = \frac{\Sigma_L \sum_{i=1}^{n_r} e_{ir}}{\hat{V} + n_r \Sigma_L}, \tag{3}$$

$$\sigma_{r|r}^L = \frac{\Sigma_L \hat{V}}{\hat{V} + n_r \Sigma_L}. \tag{4}$$

Next, the posteriors $\phi_r^j|\{y_{ir'}, i = 1, \cdots, n_{r'}, \forall r' \prec r\} \sim N(\phi_{r|r}^j, \sigma_{r|r}^j)$, are recursively updated from $j = L - 1$ to $j = 1$, by regressing the parent node effect towards each child and combining information from all the children.

To provide intuition about regression step, it is useful to invert the state equation (2) and express the distribution of $\phi_{pa(r)}^{j-1}$ conditional on $\phi_r^j$. Note that

$$\phi_{pa(r)}^{j-1} = E(\phi_{pa(r)}^{j-1}|\phi_r^j) + (\phi_{pa(r)}^{j-1} - E(\phi_{pa(r)}^{j-1}|\phi_r^j)) \tag{5}$$

Simple algebra provides the conditional expectation and variance of $\phi_{pa(r)}^{j-1}|\phi_r^j$ as

$$\phi_{pa(r)}^{j-1} = B_j\phi_r^j + \psi_r^j, \tag{6}$$

where $B_j = \sum_{i=1}^{j-1}\hat\gamma_i / \sum_{i=1}^{j}\hat\gamma_i$, correlation between any two siblings at level $j$ and $\psi_r^j \sim N(0, B_j\hat\gamma_j)$.

First, a new prior is obtained for the parent node based on the current estimate of each child by plugging-in the current estimates of a child into equation (6). For the $i^{th}$ child of node $r$ (here we assume that $r$ is at level $j - 1$, and $c_i(r)$ is at level $j$),

$$\phi_{r|c_i(r)}^{j-1} = B_j\phi_{c_i(r)|c_i(r)}^j, \tag{7}$$

$$\sigma_{r|c_i(r)}^{j-1} = B_j^2\sigma_{c_i(r)|c_i(r)}^j + B_j\hat\gamma_j, \tag{8}$$

Next, we combine information obtained by the parent from all its children.

$$\phi_{r|r}^{j-1} = \sigma_{r|r}^{j-1}\sum_{i=1}^{k_r}(\phi_{r|c_i(r)}^{j-1}/\sigma_{r|c_i(r)}^{j-1}), \tag{9}$$

$$1/\sigma_{r|r}^{j-1} = \Sigma_{j-1}^{-1} + \sum_{i=1}^{k_r}((1/\sigma_{r|c_i(r)}^{j-1}) - \Sigma_{j-1}^{-1}). \tag{10}$$

where $k_r$ is the number of children of node $r$ at level $j - 1$.

*Smoothing:*
In the smoothing step, parents propagate information recursively from root to the leaves to provide us with the posterior of each $\phi_r^j$ based on the entire data. Denoting the posterior mean and variance of $\phi_r^j$ given all the observations by $\hat\phi_r^j$ and $\sigma_r^j$ respectively, the update equations are given below.

For level 1 nodes, set $\hat\phi_r^1 = \phi_{r|r}^1$, and $\sigma_r^1 = \sigma_{r|r}^1$.

For node $r$ at other levels,

$$\hat\phi_r^j = \phi_{r|r}^j + \sigma_{r|r}^j B_j(\hat\phi_{pa(r)}^{j-1} - \phi_{pa(r)|r}^{j-1})/\sigma_{pa(r)|r}^j, \tag{11}$$

$$\sigma_r^j = \sigma_{r|r}^j + \sigma_{r|r}^{j^2}B_j^2(\sigma_{pa(r)}^{j-1} - \sigma_{pa(r)|r}^{j-1})/\sigma_{pa(r)|r}^{j^2}, \tag{12}$$

and let

$$\sigma_{r,pa(r)}^{j,j-1} = \sigma_{r|r}^j B_j\sigma_{pa(r)}^{j-1}/\sigma_{pa(r)|r}^{j-1}. \tag{13}$$

The computational complexity of the algorithm is linear in the number of nodes in the hierarchy and for each parent node, we perform an operation which is cubic in the number of children. Hence, for most hierarchies that arise in practical applications, the complexity is "essentially" linear in the number of nodes.

*Expectation Maximization:*

To estimate all parameters simultaneously, we use an EM algorithm which assumes the $\phi$ parameters to be the missing latent variables. The expectation step consists of computing the expected value of complete log-posterior with respect to the conditional distribution of missing data $\phi$, obtained using the multi-scale Kalman filter algorithm. The maximization step obtains revised estimates of other parameters by maximizing the expected complete log-posterior.

$$\hat{V} = \sum_{r \in T_L} \frac{\sum_{i=1}^{n_r} (e_{ir} - \hat{\phi}_r^L)^2 + n_r \sigma_r^L}{\sum_{r \in T_L} n_r}, \tag{14}$$

For $j = 1, \cdots, L$,

$$\hat{\gamma}_j = \frac{\sum_{r \in T_j} (\sigma_r^j + \sigma_{pa(r)}^{j-1} - 2\sigma_{r,pa(r)}^{j,j-1} + (\hat{\phi}_r^j - \hat{\phi}_{pa(r)}^{j-1})^2)}{|m_j|}. \tag{15}$$

*Updating $\hat{\boldsymbol{\beta}}$:*

We use the posterior mean of $\boldsymbol{\phi}$ obtained from the Kalman filtering step, to compute the posterior mean of $\boldsymbol{\beta}$ as given in equation (16).

$$\hat{\boldsymbol{\beta}} = (\boldsymbol{X}'\boldsymbol{X})^{-1}\boldsymbol{X}'(\boldsymbol{Y} - \hat{\boldsymbol{\phi}}^L), \tag{16}$$

where $\hat{\boldsymbol{\phi}}^L$ is the vector of $\hat{\phi}_r^L$ corresponding to each observation $y_{ir}$ at different leaf node $r$.

## 2.2 Simulation Performance

We first perform a simulation study with a hierarchy described in [7, 8]. The data focus on 2449 Guatemalan children who belong to 1558 families who in turn live in 161 communities. The response variable of interest is binary with a positive label assigned to a child if he/she received a full set of immunizations. The actual data contains 15 covariates capturing individual, family and community level characteristics as shown in Table 2. For our simulation study, we consider only three covariates, with the coefficient vector $\boldsymbol{\beta}$ set with entries all equal to 1. We simulated Gaussian response as follows: $y_{ir}|\boldsymbol{b} \sim N(\boldsymbol{x}'_{ir}\boldsymbol{\beta} + b_r^1 + b_r^2, 10)$ where $b_r^1 \sim N(0, 4)$, and $b_r^2 \sim N(0, 1)$. We simulated 100 data sets and compared the estimates from Kalman filter to the one obtained from standard routine *lme4* in the statistical software $\boldsymbol{R}$. Results from our procedure agreed almost exactly with those obtained from *lme4*, our computations was many times faster than *lme4*. The EM method converged rapidly and required at most 30 iterations.

## 3 MLH for Non-Gaussian Responses

We discuss model fitting for Bernoulli response but note that other distributions in the generalized linear model family can be easily fitted using the procedure. Let $y_{ir} \sim Bernoulli(p_{ir})$, i.e. $P(y_{ir}) = p_{ir}^{y_{ir}}(1 - p_{ir})^{1-y_{ir}}$. Let $\theta_{ir} = log\frac{p_{ir}}{1-p_{ir}}$ be the log-odds. The MLH logistic regression is defined as:

$$\theta_{ir} = \boldsymbol{x}'_{ir}\boldsymbol{\beta} + \phi_r^L, \tag{17}$$

with the same multi-level prior as described in equation (2). The non-conjugacy of the normal multi-level prior makes the computation more difficult. We take recourse to Taylor series approximation coupled with the Kalman filter algorithm. The estimates obtained are biased; we recommend cross-validation and parametric bootstrap (adapted from [4]) to correct for the bias. The bootstrap procedure though expensive is easily parallelizable and accurate.

### 3.1 Approximation Methods

Let $\eta_{ir} = \boldsymbol{x}_{ir}\hat{\boldsymbol{\beta}} + \hat{\phi}_r^L$, where $\hat{\boldsymbol{\beta}}$, $\hat{\phi}_r^L$ are current estimates of the parameters in our algorithm. We do a quadratic approximation of the log-likelihood through a second order Taylor expansion (Laplace approximation) around $\eta_{ir}$. This enables us to do the calculations as in the Gaussian case with the response $y_{ir}$ being replaced by $Z_{ir}$ where

$$Z_{ir} = \eta_{ir} + \frac{2y_{ir} - 1}{g((2y_{ir} - 1)\eta_{ir})}, \tag{18}$$

---
**Algorithm 1** The bootstrap procedure
---
Let $\boldsymbol{\theta} = (\boldsymbol{\beta}, \boldsymbol{\gamma})$.
Obtain $\tilde{\boldsymbol{\theta}}$ as an initial estimate of $\boldsymbol{\theta}$. Bias $\boldsymbol{b}^{(0)} = 0$.
**for** $i = 1$ **to** $N$ **do**
   $\hat{\boldsymbol{\theta}} = \tilde{\boldsymbol{\theta}} - \boldsymbol{b}^{(i)}$.
   **for** $j = 1$ **to** $M$ **do**
      Use $\hat{\boldsymbol{\theta}}$ to simulate new data $j$, by simulating $\phi$ and the corresponding $\boldsymbol{Y}$.
      For data $j$, obtain an new estimate of $\boldsymbol{\theta}$ as $\tilde{\boldsymbol{\theta}}^{(j)}$.
   **end for**
   $\boldsymbol{b}^{(i+1)} = \frac{1}{M} \sum\limits_{j=1}^{M} \tilde{\boldsymbol{\theta}}^{(j)} - \hat{\boldsymbol{\theta}}$.
**end for**
---

and $g(x) = 1/(1 + \exp(-x))$. Approximately,

$$Z_{ir} \sim N(\boldsymbol{x}'_{ir}\boldsymbol{\beta} + \phi_r^L, \frac{1}{g(\eta_{ir})g(-\eta_{ir})}). \tag{19}$$

Now denote $e_{ir} = Z_{ir} - \boldsymbol{x}'_{ir}\hat{\boldsymbol{\beta}}$, and the approximated variance of $Z_{ir}$ as $V_{ir}$. Analogous to equation (3) and (4), the resulting filtering step for the leaf nodes becomes:

$$\phi_{r|r}^L = \sigma_{r|r}^L \sum_{i=1}^{n_r} \frac{e_{ir}}{V_{ir}}, \tag{20}$$

$$\sigma_{r|r}^L = (\frac{1}{\Sigma_L} + \sum_{i=1}^{n_r} \frac{1}{V_{ir}})^{-1}. \tag{21}$$

The step for estimating $\boldsymbol{\beta}$ becomes:

$$\hat{\boldsymbol{\beta}} = (\boldsymbol{X}'\boldsymbol{W}\boldsymbol{X})^{-1}\boldsymbol{X}'\boldsymbol{W}(\boldsymbol{Z} - \hat{\boldsymbol{\phi}}^L), \tag{22}$$

where $\boldsymbol{W} = diag(\frac{1}{V_{ir}})$. All the other computational steps remain the same as in the Gaussian case.

### 3.2 Bias correction

Table 2 shows estimates of parameters obtained from our approximation method in the column titled $KF$. Compared to the unbiased estimates obtained from the slow Gibbs sampler, it is clear our estimates are biased. Our bias correction procedure is described in Algorithm 1. In general, a value of $M = 50$ with about $100 - 200$ iterations worked well for us. The bias corrected estimates are reported under KF-B in Table 2. The estimates after bootstrap correction are closer to the estimates obtained from Gibbs sampling. It is also customary to estimate hyper parameters like the $\boldsymbol{\gamma}$ using a tuning dataset. To test the performance of such a strategy, we created a two-dimensional grid for $(\sqrt{\gamma_1}, \sqrt{\gamma_2})$ for the epidemiological Guatemalan data set ranging in $[.1, 3] \times [.1, 3]$ and computed the log-likelihood on a $10\%$ randomly sampled hold-out data. For each point on the two-dimensional grid, we estimated the other parameters $\phi$ and $\boldsymbol{\beta}$, using our EM algorithm that does not update the value of $\boldsymbol{\gamma}$. The estimates at the optimal value of $\boldsymbol{\gamma}$ are shown in Table 2 under KF-C. The estimates are better than KF but worse than KF-B.

Based on our findings, we recommend KF-B when computing resources are available (especially multiple processors) and running time is not a big constraint; if runtime is an issue we recommend grid search using a small number of points around the initial estimate.

## 4   Content Match Data Analysis

We analyze data from an internet advertising application where every showing of an ad on a web page (called an *impression*) constitutes an event. The goal is to rank ads on a given page based on click-through rates. Building a predictive model for click-rates via features derived from pages and

| Effects | KF | KF-B | KF-C | Gibbs |
|---|---|---|---|---|
| *Fixed effects* | | | | |
| Individual | | | | |
| Child age $\geq$ 2 years | 0.99 | 1.77 | 1.18 | 1.84 |
| Mother age $\geq$ 25 years | -0.09 | -0.16 | -0.10 | -0.26 |
| Birth order 2-3 | -0.10 | -0.18 | -0.25 | -0.29 |
| Birth order 4-6 | 0.13 | 0.25 | 0.10 | 0.21 |
| Birth order $\geq$ 7 | 0.20 | 0.36 | 0.21 | 0.50 |
| Family | | | | |
| Indigenous, no Spanish | -0.05 | -0.11 | 0.02 | -0.22 |
| Indigenous Spanish | 0.00 | 0.01 | 0.02 | -0.11 |
| Mother's education primary | 0.22 | 0.44 | 0.32 | 0.48 |
| Mother's education secondary or better | 0.23 | 0.44 | 0.27 | 0.46 |
| Husband's education primary | 0.30 | 0.53 | 0.39 | 0.59 |
| Husband's education secondary or better | 0.27 | 0.48 | 0.35 | 0.55 |
| Husband's education missing | 0.02 | 0.04 | -0.08 | 0.00 |
| Mother ever worked | 0.21 | 0.35 | 0.24 | 0.42 |
| Community | | | | |
| Rural | -0.50 | -0.91 | -0.62 | -0.96 |
| Proportion indigenous, 1981 | -0.67 | -1.23 | -0.89 | -1.22 |
| | | | | |
| *Random effects* | | | | |
| Standard deviations $\gamma$ | | | | |
| Family | 0.74 | 2.40 | 1.92 | 2.60 |
| Community | 0.56 | 1.05 | 0.81 | 1.13 |

Table 2: Estimates for the binary MLH model of complete immunization (Kalman Filtering results)

ads is an attractive approach. In our case, semantic features are obtained by classifying pages and ads into a large seven-level content hierarchy that is manually constructed by humans. We form a new hierarchy (a pyramid) by taking the cross product of the two hierarchies. This is used to estimate smooth click-rates of (page,ad) pairs.

## 4.1 Training and Test Data

Although the page and ad hierarchies consist of 7 levels, classification is often done at coarser levels by the classifier. In fact, the average level at which classification took place is 3.8. To train our model, we only consider the top 3 levels of the original hierarchy. Pages and ads that are classified at coarser levels are randomly assigned to the children nodes. Overall, the pyramid has 441, 25751 and 241292 nodes for the top 3 levels. The training data were collected by confining to a specific subset of data which is sufficient to illustrate our methodology but in no way representative of the actual publisher traffic received by the ad-network under consideration. The training data we collected spans 23 days and consisted of approximately 11M binary observations with approximately 1.9M clicks. The test set consisted of 1 day's worth of data with approximately .5M observations. We randomly split the test data into 20 equal sized partitions to report our results. The covariates include the position at which an ad is shown; ranking ads on pages after adjusting for positional effects is important since the positional effects introduce strong bias in the estimates In the training data a large fraction of leaf nodes in the pyramid (approx 95%) have zero clicks, this provides a good motivation to fit the binary MLH on this data to get smoother estimates at leaf nodes by using information at coarser resolutions.

## 4.2 Results

We compare the following models using log-likelihood on the test data: a) The model which predicts a constant probability for all examples, b) 3 level MLH but without positional effects, c) top 2 level MLH to illustrate the gains of using information at a finer resolution, and d) 3 level MLH with positional effects to illustrate the generality of the approach; one can incorporate both additional features and the hierarchy into a single model. Figure 1 shows the distribution of average test likelihood on the partitions. As expected, all variations of MLH are better than the constant model. The MLH model which uses only 2 levels is inferior to the 3 level MLH while the general model that uses both covariates and hierarchy is the best.

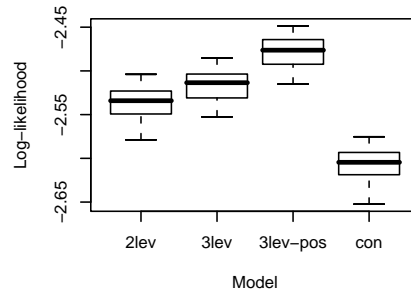

Figure 1: Distribution of test log-likelihood on 20 equal sized splits of test data.

## 5 Discussion

In applications where data is aggregated at multiple resolutions with sparsity at finer resolutions, multi-level hierarchical models provide an attractive class to reduce variance by smoothing estimates at finer resolutions using data at coarser resolutions. However, the smoothing provides a better bias-variance tradeoff only when the hierarchy provides a natural clustering for the response variable and captures some latent characteristics of the process; often true in practice. We proposed a fast novel algorithm to fit these models based on a multi-scale Kalman filter that is both scalable and easy to implement. For the non-Gaussian case, the estimates are biased but performance can be improved by using a bootstrap correction or estimation through a tuning set. In future work, we will report on models that generalize our approach to arbitrary number of hierarchies that may all have different structure. This is a challenging problem since in general cross-product of trees is not a hierarchy but a graph.

## References

[1] D. Agarwal, A. Broder, D. Chakrabarti, D. Diklic, V. Josifovski, and M. Sayyadian. Estimating rates of rare events at multiple resolutions. In *KDD*, pages 16–25, 2007.

[2] K. C. Chou, A. S. Willsky, and R. Nikoukhah. Multiscale systems, Kalman filters, and Ricatti equations. *IEEE Transactions on Automatic Control*, 39:479–492, 1994.

[3] A. Gelman and J. Hill. *Data Analysis sing Regression and Multi-Level/Hierarchical Models*. Cambridge University Press, 2007.

[4] A. Y. C. Kuk. Asymptotically unbiased estimation in generalized linear models with random effects. *Journal of the Royal Statistical Society, Series B (Methodological),*, 57:395–407, 1995.

[5] J. C. Pinheiro and D. M. Bates. *Mixed-Effects Models in S and S-PLUS*. Springer-Verlag, New York, 2000.

[6] S. W. Raudenbush, M. L. Yang, and M. Yosef. Maximum likelihood for generalized linear models with nested random effects via high-order, multivariate Laplace approximation. *Journal of Computational and Graphical Statistics*, 9(1):141–157, 2000.

[7] G. Rodriguez and N. Goldman. An assessment of estimation procedures for multilevel models with binary responses. *Journal of Royal Statistical Society, Series A,*, 158:73–89, 1995.

[8] G. Rodriguez and N. Goldman. Improved estimation procedures for multilevel models with binary response: A case-study. *Journal of the Royal Statistical Society, Series A,*, 164(2):339–355, 2001.

[9] S. K. Sahu and A. E. Gelfand. Identifiability, improper Priors, and Gibbs sampling for generalized linear models. *Journal of the American Statistical Association*, 94(445):247–254, 1999.

